# Classification by Pairwise Coupling

TREVOR HASTIE *
Stanford University
and
ROBERT TIBSHIRANI †
University of Toronto

## Abstract

We discuss a strategy for polychotomous classification that involves estimating class probabilities for each pair of classes, and then coupling the estimates together. The coupling model is similar to the Bradley-Terry method for paired comparisons. We study the nature of the class probability estimates that arise, and examine the performance of the procedure in simulated datasets. The classifiers used include linear discriminants and nearest neighbors: application to support vector machines is also briefly described.

## 1  Introduction

We consider the discrimination problem with $K$ classes and $N$ training observations. The training observations consist of predictor measurements $\mathbf{x} = (x_1, x_2, \ldots x_p)$ on $p$ predictors and the known class memberships. Our goal is to predict the class membership of an observation with predictor vector $\mathbf{x}_0$

Typically $K$-class classification rules tend to be easier to learn for $K = 2$ than for $K > 2$ — only one decision boundary requires attention. Friedman (1996) suggested the following approach for the the $K$-class problem: solve each of the two-class problems, and then for a test observation, combine all the pairwise decisions to form a $K$-class decision. Friedman's combination rule is quite intuitive: assign to the class that wins the most pairwise comparisons.

Department of Statistics, Stanford University, Stanford California 94305; trevor@playfair.stanford.edu

Department of Preventive Medicine and Biostatistics, and Department of Statistics; tibs@utstat.toronto.edu

Friedman points out that this rule is equivalent to the Bayes rule when the class posterior probabilities $p_i$ (at the test point) are known:

$$\text{argmax}_i[p_i] = \text{argmax}_i[\sum_{j \neq i} I\left(p_i/(p_i + p_j) > p_j/(p_i + p_j)\right)]$$

Note that Friedman's rule requires only an estimate of each pairwise decision. Many (pairwise) classifiers provide not only a rule, but estimated class probabilities as well. In this paper we argue that one can improve on Friedman's procedure by combining the pairwise class probability estimates into a joint probability estimate for all $K$ classes.

This leads us to consider the following problem. Given a set of events $A_1, A_2, \ldots A_K$, some experts give us pairwise probabilities $r_{ij} = \text{Prob}(A_i|A_i \text{ or } A_j)$. Is there a set of probabilities $p_i = \text{Prob}(A_i)$ that are compatible with the $r_{ij}$?

In an exact sense, the answer is no. Since $\text{Prob}(A_i|A_i \text{ or } A_j) = p_j/(p_i + p_j)$ and $\sum p_i = 1$, we are requiring that $K-1$ free parameters satisfy $K(K-1)/2$ constraints and, this will not have a solution in general. For example, if the $r_{ij}$ are the $ij$th entries in the matrix

$$\begin{pmatrix} \cdot & 0.9 & 0.4 \\ 0.1 & \cdot & 0.7 \\ 0.6 & 0.3 & \cdot \end{pmatrix} \tag{1}$$

then they are not compatible with any $p_i$'s. This is clear since $r_{12} > .5$ and $r_{23} > .5$, but also $r_{31} > .5$.

The model $\text{Prob}(A_i|A_i \text{ or } A_j) = p_j/(p_i + p_j)$ forms the basis for the Bradley-Terry model for paired comparisons (Bradley & Terry 1952). In this paper we fit this model by minimizing a Kullback-Leibler distance criterion to find the best approximation $\hat{r}_{ij} = \hat{p}_i/(\hat{p}_i + \hat{p}_j)$ to a given set of $r_{ij}$'s. We carry this out at each predictor value $\mathbf{x}$, and use the estimated probabilities to predict class membership at $\mathbf{x}$.

In the example above, the solution is $\hat{\mathbf{p}} = (0.47, 0.25, 0.28)$. This solution makes qualitative sense since event $A_1$ "beats" $A_2$ by a larger margin than the winner of any of the other pairwise matches.

Figure 1 shows an example of these procedures in action. There are 600 data points in three classes, each class generated from a mixture of Gaussians. A linear discriminant model was fit to each pair of classes, giving pairwise probability estimates $r_{ij}$ at each $\mathbf{x}$. The first panel shows Friedman's procedure applied to the pairwise rules. The shaded regions are areas of indecision, where each class wins one vote. The coupling procedure described in the next section was then applied, giving class probability estimates $\hat{\mathbf{p}}(\mathbf{x})$ at each $\mathbf{x}$. The decision boundaries resulting from these probabilities are shown in the second panel. The procedure has done a reasonable job of resolving the confusion, in this case producing decision boundaries similar to the three-class LDA boundaries shown in panel 3. The numbers in parentheses above the plots are test-error rates based on a large test sample from the same population. Notice that despite the indeterminacy, the max-wins procedure performs no worse than the coupling procedure. and both perform better than LDA. Later we show an example where the coupling procedure does substantially better than max-wins.

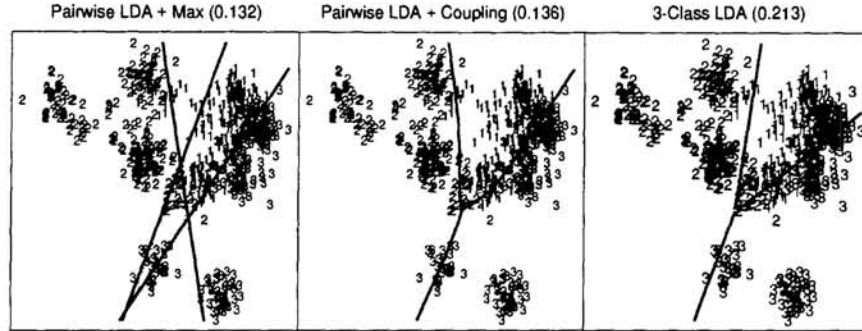

Figure 1: *A three class problem, with the data in each class generated from a mixture of Gaussians. The first panel shows the maximum-win procedure. The second panel shows the decision boundary from coupling of the pairwise linear discriminant rules based on $\hat{d}$ in (6). The third panel shows the three-class LDA boundaries. Test-error rates are shown in parentheses.*

This paper is organized as follows. The coupling model and algorithm are given in section 2. Pairwise threshold optimization, a key advantage of the pairwise approach, is discussed in section 3. In that section we also examine the performance of the various methods on some simulated problems, using both linear discriminant and nearest neighbour rules. The final section contains some discussion.

## 2   Coupling the probabilities

Let the probabilities at feature vector $\mathbf{x}$ be $\mathbf{p}(\mathbf{x}) = (p_1(\mathbf{x}), \ldots p_K(\mathbf{x}))$. In this section we drop the argument $\mathbf{x}$, since the calculations are done at each $\mathbf{x}$ separately.

We assume that for each $i \neq j$, there are $n_{ij}$ observations in the training set and from these we have estimated conditional probabilities $r_{ij} = \text{Prob}(i|i \text{ or } j)$.

Our model is

$$\begin{aligned} n_{ij}r_{ij} &\sim \text{Binomial}(n_{ij}, \mu_{ij}) \\ \mu_{ij} &= \frac{p_i}{p_i + p_j} \end{aligned} \qquad (2)$$

or equivalently

$$\log \mu_{ij} = \log(p_i) - \log(p_i + p_j), \qquad (3)$$

a *log-nonlinear model*.

We wish to find $\hat{p}_i$'s so that the $\hat{u}_{ij}$'s are close to the $r_{ij}$'s. There are $K-1$ independent parameters but $K(K-1)/2$ equations, so it is not possible in general to find $\hat{p}_i$'s so that $\hat{\mu}_{ij} = r_{ij}$ for all $i, j$.

Therefore we must settle for $\hat{\mu}_{ij}$'s that are close to the observed $r_{ij}$'s. Our closeness criterion is the average (weighted) Kullback-Leibler distance between $r_{ij}$ and $\mu_{ij}$:

$$\ell(\mathbf{p}) = \sum_{i<j} n_{ij} \left[ r_{ij} \log \frac{r_{ij}}{\mu_{ij}} + (1 - r_{ij}) \log \frac{1 - r_{ij}}{1 - \mu_{ij}} \right] \qquad (4)$$

and we find **p** to minimize this function.

This model and criterion is formally equivalent to the Bradley-Terry model for preference data. One observes a proportion $r_{ij}$ of $n_{ij}$ preferences for item $i$, and the sampling model is binomial, as in (2). If each of the $r_{ij}$ were independent, then $\ell(\mathbf{p})$ would be equivalent to the log-likelihood under this model. However our $r_{ij}$ are not independent as they share a common training set and were obtained from a common set of classifiers. Furthermore the binomial models do not apply in this case; the $r_{ij}$ are evaluations of functions at a point, and the randomness arises in the way these functions are constructed from the training data. We include the $n_{ij}$ as weights in (4); this is a crude way of accounting for the different precisions in the pairwise probability estimates.

The score (gradient) equations are:

$$\sum_{j \neq i} n_{ij}\mu_{ij} = \sum_{j \neq i} n_{ij}r_{ij}; \; i = 1, 2, \ldots K \tag{5}$$

subject to $\sum p_i = 1$. We use the following iterative procedure to compute the $\hat{p}_i$'s:

## Algorithm

1. Start with some guess for the $\hat{p}_i$, and corresponding $\hat{\mu}_{ij}$.

2. Repeat $(i = 1, 2, \ldots, K, 1, \ldots)$ until convergence:

$$\hat{p}_i \leftarrow \hat{p}_i \cdot \frac{\sum_{j \neq i} n_{ij}r_{ij}}{\sum_{j \neq i} n_{ij}\hat{\mu}_{ij}}$$

   renormalize the $\hat{p}_i$, and recompute the $\hat{\mu}_{ij}$.

3. $\hat{\mathbf{p}} \leftarrow \hat{\mathbf{p}}/\sum \hat{p}_i$

The algorithm also appears in Bradley & Terry (1952). The updates in step 2 attempt to modify **p** so that the sufficient statistics match their expectation, but go only part of the way. We prove in Hastie & Tibshirani (1996) that $\ell(\mathbf{p})$ increases at each step. Since $\ell(\mathbf{p})$ is bounded above by zero, the procedure converges. At convergence, the score equations are satisfied, and the $\hat{\mu}_{ij}$s and $\hat{\mathbf{p}}$ are consistent. This algorithm is similar in flavour to the Iterative Proportional Scaling (IPS) procedure used in log-linear models. IPS has a long history, dating back to Deming & Stephan (1940). Bishop, Fienberg & Holland (1975) give a modern treatment and many references.

The resulting classification rule is

$$\dot{d}(\mathbf{x}) = \text{argmax}_i[\hat{p}_i(\mathbf{x})] \tag{6}$$

Figure 2 shows another example similar to Figure 1, where we can compare the performance of the rules $\hat{d}$ and $\tilde{d}$. The hatched area in the top left panel is an indeterminate region where there is more than one class achieving $\max(\tilde{p}_i)$. In the top right panel the coupling procedure has resolved this indeterminacy in favor of class 1 by weighting the various probabilities. See the figure caption for a description of the bottom panels.

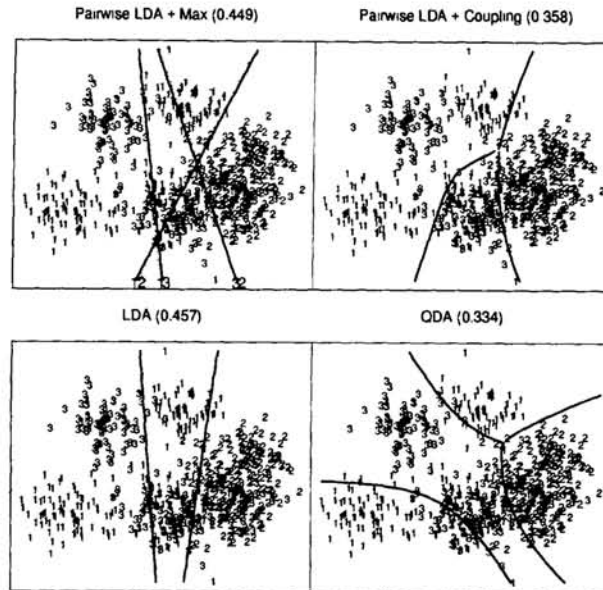

Figure 2: *A three class problem similar to that in figure 1, with the data in each class generated from a mixture of Gaussians. The first panel shows the maximum-wins procedure d̃). The second panel shows the decision boundary from coupling of the pairwise linear discriminant rules based on d̃ in (6). The third panel shows the three-class LDA boundaries, and the fourth the QDA boundaries. The numbers in the captions are the error rates based on a large test set from the same population.*

## 3   Pairwise threshold optimization

As pointed out by Friedman (1996), approaching the classification problem in a pairwise fashion allows one to optimize the classifier in a way that would be computationally burdensome for a $K$-class classifier. Here we discuss optimization of the classification threshold.

For each two class problem, let logit $\mathrm{p_{ij}(x)} = \mathrm{d_{ij}(x)}$. Normally we would classify to class $i$ if $d_{ij}(x) > 0$. Suppose we find that $d_{ij}(x) > t_{ij}$ is better. Then we define $d'_{ij}(x) = d_{ij}(x) - t_{ij}$, and hence $p'_{ij}(x) = \mathrm{logit}^{-1}\mathrm{d'_{ij}(x)}$. We do this for all pairs, and then apply the coupling algorithm to the $p'_{ij}(x)$ to obtain probabilities $p'_i(x)$. In this way we can optimize over $K(K-1)/2$ parameters separately, rather than optimize jointly over $K$ parameters. With nearest neigbours, there are other approaches to threshold optimization, that bias the class probability estimates in different ways. See Hastie & Tibshirani (1996) for details. An example of the benefit of threshold optimization is given next.

*Example: ten Gaussian classes with unequal covariance*

In this simulated example taken from Friedman (1996), there are 10 Gaussian classes in 20 dimensions. The mean vectors of each class were chosen as 20 independent uniform $[0, 1]$ random variables. The covariance matrices are constructed from eigenvectors whose square roots are uniformly distributed on the 20-dimensional unit sphere (subject to being mutually orthogonal), and eigenvalues uniform on $[0.01, 1.01]$. There are 100 observations per class in the training set, and 200 per

class in the test set. The optimal decision boundaries in this problem are quadratic, and neither linear nor nearest-neighbor methods are well-suited. Friedman states that the Bayes error rate is less than 1%.

Figure 3 shows the test error rates for linear discriminant analysis, $J$-nearest neighbor and their paired versions using threshold optimization. We see that the coupled classifiers nearly halve the error rates in each case. In addition, the coupled rule works a little better than Friedman's max rule in each task. Friedman (1996) reports a median test error rate of about 16% for his thresholded version of pairwise nearest neighbor.

Why does the pairwise thresholding work in this example? We looked more closely at the pairwise nearest neighbour rules rules that were constructed for this problem. The thresholding biased the pairwise distances by about 7% on average. The average number of nearest neighbours used per class was 4.47 (.122), while the standard $J$-nearest neighbour approach used 6.70 (.590) neighbours for all ten classes. For all ten classes, the 4.47 translates into 44.7 neighbours. Hence relative to the standard $J$-NN rule, the pairwise rule, in using the threshold optimization to reduce bias, is able to use about six times as many near neighbours.

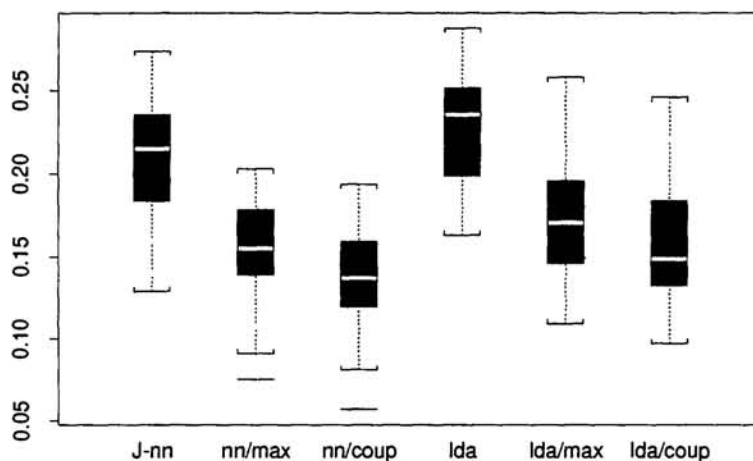

Figure 3: *Test errors for 20 simulations of ten-class Gaussian example.*

## 4  Discussion

Due to lack of space, there are a number of issues that we did not discuss here. In Hastie & Tibshirani (1996), we show the relationship between the pairwise coupling and the max-wins rule: specifically, if the classifiers return 0 or 1s rather than probabilities, the two rules give the same classification. We also apply the pairwise coupling procedure to nearest neighbour and support vector machines. In the latter case, this provides a natural way of extending support vector machines, which are defined for two-class problems, to multi-class problems.

The pairwise procedures, both Friedman's max-win and our coupling, are most likely to offer improvements when additional optimization or efficiency gains are possible in the simpler 2-class scenarios. In some situations they perform exactly like the multiple class classifiers. Two examples are: a) each of the pairwise rules are based on QDA: i.e. each class modelled by a Gaussian distribution with separate covariances, and then the $r_{ij}$s derived from Bayes rule; b) a generalization of the above, where the density in each class is modelled in some fashion, perhaps nonparametrically via density estimates or near-neighbor methods, and then the density estimates are used in Bayes rule.

Pairwise LDA followed by coupling seems to offer a nice compromise between LDA and QDA, although the decision boundaries are no longer linear. For this special case one might derive a different coupling procedure globally on the *logit* scale, which would guarantee linear decision boundaries. Work of this nature is currently in progress with Jerry Friedman.

## Acknowledgments

We thank Jerry Friedman for sharing a preprint of his pairwise classification paper with us, and acknowledge helpful discussions with Jerry, Geoff Hinton, Radford Neal and David Tritchler. Trevor Hastie was partially supported by grant DMS-9504495 from the National Science Foundation, and grant ROI-CA-72028-01 from the National Institutes of Health. Rob Tibshirani was supported by the Natural Sciences and Engineering Research Council of Canada and the IRIS Centr of Excellence.

# References

Bishop, Y., Fienberg, S. & Holland, P. (1975), *Discrete multivariate analysis*, MIT Press, Cambridge.

Bradley, R. & Terry, M. (1952), 'The rank analysis of incomplete block designs. i. the method of paired comparisons', *Biometrics* pp. 324–345.

Deming, W. & Stephan, F. (1940), 'On a least squares adjustment of a sampled frequency table when the expected marginal totals are known', *Ann. Math. Statist.* pp. 427–444.

Friedman, J. (1996), Another approach to polychotomous classification, Technical report, Stanford University.

Hastie, T. & Tibshirani, R. (1996), Classification by pairwise coupling, Technical report, University of Toronto.